# Unsupervised Structure Discovery for Semantic Analysis of Audio

**Sourish Chaudhuri**
Language Technologies Institute
Carnegie Mellon University
Pittsburgh, PA 15213
sourishc@cs.cmu.edu

**Bhiksha Raj**
Language Technologies Institute
Carnegie Mellon University
Pittsburgh, PA 15213
bhiksha@cs.cmu.edu

## Abstract

Approaches to audio classification and retrieval tasks largely rely on detection-based discriminative models. We submit that such models make a simplistic assumption in mapping acoustics directly to semantics, whereas the actual process is likely more complex. We present a generative model that maps acoustics in a hierarchical manner to increasingly higher-level semantics. Our model has two layers with the first layer modeling generalized sound units with no clear semantic associations, while the second layer models local patterns over these sound units. We evaluate our model on a large-scale retrieval task from TRECVID 2011, and report significant improvements over standard baselines.

## 1 Introduction

Automatic semantic analysis of multimedia content has been an active area of research due to potential implications for indexing and retrieval [1–7]. In this paper, we limit ourselves to the analysis of the audio component of multimedia data only. Early approaches for semantic indexing of audio relied on automatic speech recognition techniques to generate semantically relevant keywords [2]. Subsequently, supervised approaches were developed for detecting specific (potentially semantically relevant) sounds in audio streams [6, 8–10], *e.g.* gunshots, laughter, music, crowd sounds etc., and using the detected sounds to characterize the audio files. While this approach has been shown to be effective on certain datasets, it requires data for ***each*** of the various sounds expected in the dataset. Further, such detectors will not generalize across datasets with varying characteristics; *e.g.* audio libraries are studio-quality, while user-generated Youtube-style content are noisy.

In order to avoid the issues that arise with using supervised, detection-based systems, unsupervised approaches were developed to learn sound dictionaries from the data [7, 11, 12],. Typically, these methods use clustering techniques on fixed length audio segments to learn a dictionary, and then characterize new data using this dictionary. However, characterizing audio data with elements from an audio dictionary (supervised or unsupervised) for semantic analysis involves an implicit assumption that the acoustics map directly to semantics. In reality, we expect the mapping to be more complex, because acoustically similar sounds can be produced by very different sources. Thus, to accurately identify the underlying semantics, we would need to effectively use more (and perhaps, deeper) structure, such as the sound context, while making inferences.

In this paper, we present a novel hierarchical, generative framework that can be used for deeper analysis of audio, and which attempts to model the underlying process that humans use in analyzing audio. Further, since most audio datasets do not contain detailed hierarchical labels that our framework would require, we present unsupervised formulations for two layers in this hierarchical framework, building on previous work for the first layer, and developing a model for the second.

However, since detailed annotations are not available, we cannot directly evaluate the induced structure on test data. Instead, we use features derived from this structure to characterize audio, and evaluate these characterizations in a large-scale audio retrieval task with semantic categories, where our model significantly improves over state-of-the-art baselines. A further benefit of the induced structure is that the generated segments may be used for annotation by humans, thus removing the need for the annotator to scan the audio to identify and mark segment boundaries, making the annotation process much faster [13].

In Section 2, we introduce a novel framework for mapping acoustics to semantics for deeper analysis of audio. Section 3 describes the process of learning the lower-level acoustic units in the framework, while Section 4 describes a generative model that automatically identifies patterns over and segments these acoustic units. Section 5 describes our experiments and results, and we conclude in Section 6.

## 2 A Hierarchical Model for (Audio) Perception

The world around us is structured in space and time, and the evolution over time of naturally occurring phenomena is related to the previous states. Thus, changes in real-world scenes are sequential by nature and the human brain can perceive this sequentiality and use it to learn semantic relationships between the various events to analyze scenes; *e.g.* the movement of traffic and people at an intersection are governed by the traffic laws. In this section, we present a hierarchical model that maps observed scene characteristics to semantics in a hierarchical fashion. We present this framework (and our experiments) in the context of audio, but it should apply to other modalities (*e.g.* video), that require semantic analysis of information sequences.

Traditional detection-based approaches, that assign each frame or a sequence of frames of pre-specified length to sound categories/clusters, are severely limited in their ability to account for context. In addition to context, we need to consider the possibility of polysemy in sounds– semantically different sounds may be acoustically similar; *e.g.* a dull metallic sound may be produced by a hammer striking an object, a baseball bat hitting a ball, or a car collision. The sound alone doesn't provide us with sufficient information to infer the semantic context. However, if the sound is followed by applause, we guess the context to be baseball, screams or sirens suggest an accident, while monotonic repetitions of the metallic sound suggest someone using a hammer. In order to automatically analyze scenes better, we need more powerful models that can handle temporal context.

In Figure 1a, we present a conceptual representation of a hierarchical framework that envisions a system to perform increasingly complex analysis of audio. The grey circles closest to the observed audio represent short-duration lower-level acoustic units which produce sounds that human ears can perceive, such as the *clink* of glass, *thump* produced by footsteps, etc. These units have acoustic characteristics, but no clear associated semantics since the semantics may be context dependent. Sequences of these units, however, will have interpretable semantics– we refer to these as *events* marked by grey rectangles in Figure 1a. The annotations in blue correspond to (usually unavailable) human labels for these events. Further, these events themselves likely influence future events, shown by the arrows, *e.g.* the loud cheering in the audio clip is because a hitter hit a home run.

Figure 1b shows the kind of structured information that we envision parsing from the audio. The lowest level, indexed by $a$, correspond to the lower-level units. The event layer in Figure 1b has been further divided into 2, where the lower level (indexed by $v$) correspond to observable events (*e.g.* hit-ball, cheering), whereas the higher level ($e$) corresponds to a semantic event (*e.g.* batting-in-run), and the root node represents the semantic category (baseball, in this case). The cost of obtaining such hierarchical annotations would be very high due to the complexity of the annotation task. Typically, audio datasets contain only a category or genre label for each audio file. As a result, models for learning such structure must be able to operate in an unsupervised framework.

This framework for semantic analysis of audio is the first effort to extract deeper semantic structure, to the best of our knowledge. In this paper, we deal only with the 2 lowest levels in Figure 1b. We build on previous work to automatically learn lower level units unsupervised from audio data [14]. We then develop a generative model to learn *event* patterns over the lower-level units, which correspond to the second layer in Figure 1b. We represent the audio as a sequence of 39-dimensional feature vectors, each comprising 13 Mel-Frequency Cepstral Coefficients and 13-dimensional$\Delta$ and $\Delta\Delta$ features.

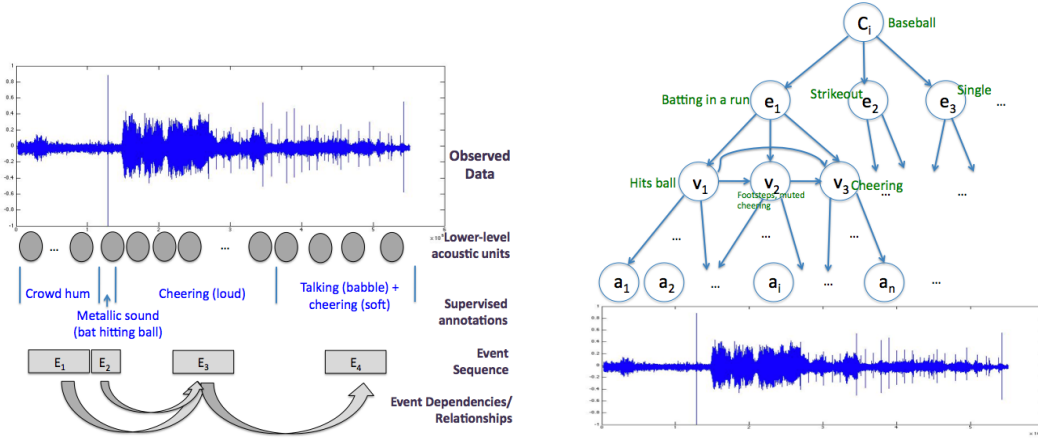

Figure 1: Conceptual representation of the proposed hierarchical framework (a) Left figure: Conceptualizing increasingly complex semantic analysis; (b) Right figure: An example semantic parse for baseball .

## 3 Unsupervised Learning of the Acoustic Unit Lexicon

At the the lowest level of the hierarchical structure specified by the model of Figure 1a is a sequence of atomic acoustic units, as described earlier. In reality, the number of such acoustic units is very large, possibly even infinite. Moreover, *annotated* training data from which they may be learned are largely unavailable.

For the task of learning a lexicon of lower-level acoustic units, we leverage an unsupervised learning framework proposed in [14], which employs a the generative model shown in Figure 2 to describe audio recordings. We define a finite set of audio symbols $\mathcal{A}$, and corresponding to each symbol $a \in \mathcal{A}$, we define an *acoustic* model $\lambda_a$, and we refer to the set of all acoustic models as $\Lambda$. According to the model, in order to generate a recording, a transcription $T$ comprising a sequence of symbols from $\mathcal{A}$ is first generated, according to a language model (LM) distribution with parameters $H$. Thereafter, for each symbol $a_t$ in $T$, a variable-length audio segment $D_{a_t}$ is generated in accordance with $\lambda_{a_t}$. The final audio $D$ comprises the concatenation of the audio segments corresponding to all the symbols in $T$. Similar to [14], we represent each acoustic unit as a 5-state HMM with gaussian mixture output densities.

The parameters of the model may be learnt using an iterative EM-algorithm shown in Algorithm 1. The learnt parameters $\lambda_a$ for each symbol $a \in \mathcal{A}$ allow us to decode any new audio file in terms of the set of symbols. While these symbols are not guaranteed to have any semantic interpretations, we expect them to capture acoustically consistent phenomena, and we see later that they do so in Figure 7. The symbols may hence be interpreted as representing generalized acoustic units (representing *clusters* of basic sound units). As in [14], we refer to these units as "Acoustic Unit Descriptors" or AUDs.

---

**Algorithm 1** Algorithm for Learning Acoustic Unit Lexicons – $(r+1)$-th iteration. $D_i$: the $i$-th audio file; $T_i$: $D_i$'s s transcript in terms of AUDs; $\Lambda$: set of AUD parameters and $H$: the LM

$$T_i^{r+1} \quad = \quad \operatorname{argmax}_T P(T|D_i; H^r; \Lambda^r) \tag{1}$$

$$\Lambda^{r+1} \quad = \quad \operatorname{argmax}_\Lambda \prod_{D_i} P(D_i|T_i^{r+1}; \Lambda) \tag{2}$$

$$H^{r+1} \quad = \quad \operatorname{argmax}_H \prod_{D_i} P(T_i^{r+1}; H) \tag{3}$$

---

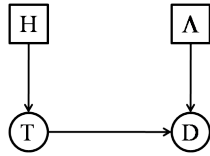

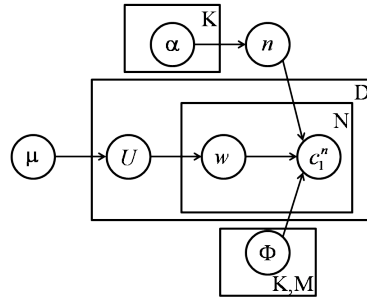

Figure 2: The generative model for generating audio from the acoustic units. $H$ and $\Lambda$ are the language model and acoustic model parameters, $T$ is the latent transcript and $D$ is the observed data.

Figure 3: The unigram based generative model for segmentation. Only $c_1^n$ is observed.

## 4  A Generative Model for Inducing Patterns over AUDs

As discussed in Section 2, we expect that audio data are composed of a sequence of semantically meaningful *events* which manifest themselves in various acoustic forms, depending on the context. The acoustic unit (AUD) lexicon described in Section 3 automatically learns the various acoustic manifestations from a dataset but do not have interpretable semantic meaning. Instead, we expect to find semantics in the local patterns over the AUDs. In this section, we introduce a generative model for the second layer in Fig 1a where the semantically interpretable acoustic events generate lower level AUDs (and thus, the observed audio).

The distribution of AUDs for a specific event will be stochastic in nature (*e.g.* segments for a *cheering* event may contain any or all of claps, shouts, speech, music), and the distribution of the events themselves are stochastic and category-dependent. Again, while the number of such events can be expected to be very large, we assume that for a given dataset, a limited number of events can describe the event space fairly well. Further, we expect that the distribution of naturally occurring events in audio will follow the power law properties typically found in natural distributions [15, 16].

We encode these intuitions in a generative model where we impose a power-law prior on the distribution of events. Events, drawn from this distribution, then generate lower level acoustic units (AUDs) corresponding to the sounds that are to be produced. Because this process is stochastic, different occurrences of the same event may produce different sequences of AUDs, which are variants of a common underlying pattern.

The generative model is shown in Figure 3. We assume $K$ audio events in the vocabulary, and $M$ distinct AUD tokens, and we can generate a corpus of $D$ documents as follows: for each document $d$, we first draw a unigram distribution $U$ for the events based on a power-law prior $\mu$. We then draw $N_d$ event tokens from the distribution for the events. Each event token can generate a sequence of AUDs of variable length $n$, where $n$ is drawn from an event specific distribution $\alpha$. $n$ AUDs ($c_1^n$) are now drawn from the multinomial AUD-emission distribution for the event $\Phi_{event}$. Thus, in this model, each audio document is a bag of events and each occurrence of an event is a bag of AUDs; the events themselves are distributions over AUDs.

At training time, only the AUD token sequences are observed. We referring to the observed AUD tokens as as $\mathcal{X}$, the latent variables as $\mathbf{Z}$ and the parameters for our process ($\mu$, $\alpha$ and $\Phi$) as $\Theta$, we can write the joint probability of all the variables in this model as shown in Equation 4. In the following subsections, we will outline a framework for training the parameter set for this model. We can then use these parameters to estimate the latent events present in audio based on an observed AUD stream (the AUD stream is obtained by decoding audio as described in Section 3).

$$P(\mathcal{X}, \mathbf{Z}, \mathbf{\Theta}) = \prod_{\mathbf{d}} \mathbf{P}(\mathbf{U_d}; \mu) \prod_{\mathbf{i}} \mathbf{P}(\mathbf{w_i^d}|\mathbf{U_d})\mathbf{P}(\mathbf{n_i^d}|\mathbf{w_i^d}; \alpha)\mathbf{P}(\mathbf{c_1^n}|\mathbf{w_i^d}, \mathbf{n_i^d}; \mathbf{\Phi}) \qquad (4)$$

This formulation of unsupervised event induction from AUD streams bears some similarities to approaches in text processing for discovering word boundaries and morphological segmentation [17–20] where a token stream is input to the system and we wish to learn models that can appro-

priately segment new sequences. Unlike those, however, we model each *event* as a bag of AUDs as opposed to an AUD sequence for two reasons. First, AUD sequences (and indeed, the observed audio) for different instances of the same event will have innate variations. Second, in the case of the audio, presence of multiple sounds may result in noisy AUD streams so that text character streams which are usually clean are not directly analogous; instead, noisy, badly spelt text might be a better analogy.

We chose the 2-parameter ($r$, $p$) Negative Binomial (Equation 5) distribution for $\alpha$, which approaches the Poisson distribution as $r$ tends to infinity, and the $r$ controls deviation from the Poisson. The power law prior is imposed by a 1-parameter ($s$) distribution shown in Equation 6 ($w^{(k)}$ represents the $k$-th most frequent word), where the parameter $s$ is drawn from $\mathcal{N}(\mu, \sigma^2)$. For English text, the value of $s$ has been observed to be very close to 1.

$$n \sim NB(r, p), s.t. \ P(n = k) = \binom{k + r - 1}{k} p^k (1 - p)^r \tag{5}$$

$$P(w^{(k)}; s, n) = \frac{\frac{1}{k^s}}{\sum_{i=1}^{i=n} \frac{1}{i^s}} \tag{6}$$

Various methods can be used for parameter learning. In Section 4.1, we present an HMM-like model that is used to estimate the parameters in an Expectation-Maximization (EM) framework [21]. Section 4.2 describes how the learning framework is used to update parameter estimates iteratively.

## 4.1 Latent Variable Estimation in the Learning Framework

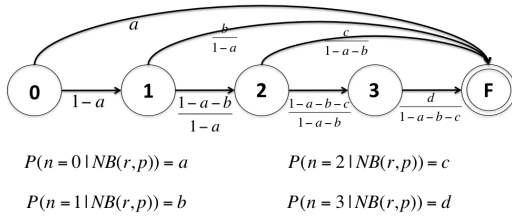

$P(n = 0 \mid NB(r, p)) = a$      $P(n = 2 \mid NB(r, p)) = c$

$P(n = 1 \mid NB(r, p)) = b$      $P(n = 3 \mid NB(r, p)) = d$

Figure 4: An example automaton for a word of maximum length 3. $a$, $b$, $c$ and $d$ represent the probabilities of lengths 0 to 3 given the parameters $r$ and $p$ for the negative binomial distribution.

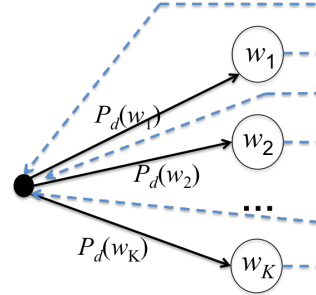

Figure 5: An automaton with the $K$ word automatons in parallel for decoding a token stream

We construct an automaton for each of the $K$ events– an example is shown in Figure 4. This example allows a maximum length of 3, and has 4 states for lengths 0 to 3 and a fifth dummy terminal state. The state for length 0[1] behaves as the start state, while F is the terminal state. An AUD is emitted whenever the automaton enters any non-final state. The transition probabilities in the automaton are governed by the negative binomial parameters for that event. Based on these, states can skip to the final state, thus accounting for variable lengths of events in terms of number of AUDs. We define $\mathcal{S}$ as the set of all start states for events, so that $\mathcal{S}_i$=start state of event $i$. Since we model event occurrences as bags of AUDs, AUD emission probabilities are shared by all states for a given event.

The automatons for the events are now put together as shown in Figure 5– the black circle represents a dummy start state, and terminal states for each event can transition to this start state. $P_d(w_i)$ represents the probability of the event $w_i$ given the unigram distribution for the document $d$. Now, given a sequence of observed tokens, we can use the automaton in Figure 5 to compute a forward table and backward table, in exactly the same manner as in HMMs. At training time, we combine the forward and backward tables to obtain our expected counts, while at test time, we can use the Viterbi algorithm to simply obtain the most likely decode for the observation sequence in terms of the latent events.

Let us refer to the forward table as $\alpha$ where $\alpha(i, t) = P(state = i, t | c_1^t)$, and let $\beta$ refer to the backward table where $\beta(i, t) = P(state = i, t | c_{t+1}^n)$. We can compute the likelihood of being in state $i$ (and extend that to being in word $i$) at time-step $t$ given the entire observation sequence:

$$P(state = i, t) \;\;=\;\; \frac{\alpha(i, t) \times \beta(i, t)}{\sum_j \alpha(j, t) \times \beta(j, t)} \tag{7}$$

$$P(w_i, t) \;\;=\;\; \frac{\sum_{k \in w_i} \alpha(k, t) \times \beta(k, t)}{\sum_j \alpha(j, t) \times \beta(j, t)} \tag{8}$$

The forward-backward tables are constructed with our current estimates and the sufficient expected counts are obtained using these estimates, which are then used to update the parameters.

## 4.2 Parameter Estimation

We obtain the EM-update equations by maximizing the (log-)likelihood from Equation 4. The forward-backward tables for each AUD stream are used to obtain the sufficient counts, as described in Section 4.1. To update the AUD emission probabilities, $\Phi_{ij}$ (AUD $j$ emitted by event $i$), we use:

$$E - Step \;\; : \;\; \sum_Z P(Z | \mathcal{X}; \Theta^r) \nu_{ij}(Z) = \sum_{t=1}^{T} P(w_i, t) \mathcal{I}(c_t = j) \tag{9}$$

$$M - step \;\; : \;\; \Phi_{ij} = \frac{\sum_d \left[ \sum_Z P(Z | \mathcal{X}; \Theta^r) \nu_{ij}(Z) \right]}{\sum_{j=1}^{M} \sum_d \left[ \sum_Z P(Z | \mathcal{X}; \Theta^r) \nu_{ij}(Z) \right]} \tag{10}$$

Here, $\nu_{ij}(Z)$ refers to the count of character $j$ emitted by word $i$ in the latent sequence $Z$. $\mathcal{I}(c_t = j)$ represents an indicator function that is 1 when the the token at time-step $t$ is $j$, and 0 otherwise.

To update the NB parameters for each event, we compute the top-$N$ paths through each training sequence in the E-step (we used $N = 50$, but ideally, $N$ should be as large as possible). Thus, if for word $i$, we have a set of $m$ occurrences in these paths of lengths $n_1, n_2, ..., n_m$, we can estimate $r$ and $p$ using Equation 11. $p$ has a closed form solution (Eqn 12) but Eqn 13 for $r$ needs an iterative numerical solution. [$\psi()$ is the digamma function]

$$L = \prod_{i=1}^{m} NB(x = n_i; r, p) \tag{11}$$

$$p = \frac{\sum_{i=1}^{m} \frac{n_i}{m}}{r + \sum_{i=1}^{m} \frac{n_i}{m}} \tag{12}$$

$$\sum_{i=1}^{m} \psi(n_i + r) - m \times \psi(r) + m \times ln\left(\frac{r}{r + \sum_{i=1}^{m} \frac{n_i}{m}}\right) = 0 \tag{13}$$

To estimate the $\mathcal{N}(\mu, \sigma^2)$ for the power-law parameter $s$, we compute expected event frequencies $E_{f_i}$ for all events for each AUD stream. This can be done using the forward-backward table as shown in Equation 14 and 15. The Zipf parameter is estimated as the slope of the best-fit line between the log-expected-frequencies ($Y$) and log-rank ($X = [log\_rank \;\; 1]^T$). The set of $s$ values in the corpus are used to estimate the $\mu$ and $\sigma^2$.

$$E - Step \;\; : \;\; count(w_i) = \sum_{t=1}^{T} P(state = \mathcal{S}_i, t) \tag{14}$$

$$E_{f_i} = \frac{count(w_i)}{\sum_j count(w_j)} \tag{15}$$

$$M - step \;\; : \;\; s_d = (Y X^+)_0, \forall d \in D \tag{16}$$

$$\mu^*, \sigma^{2*} = \arg\max_{\mu, \sigma^2} \prod_{i=1}^{i=D} P(s_i | \mathcal{N}(\mu, \sigma^2)) \tag{17}$$

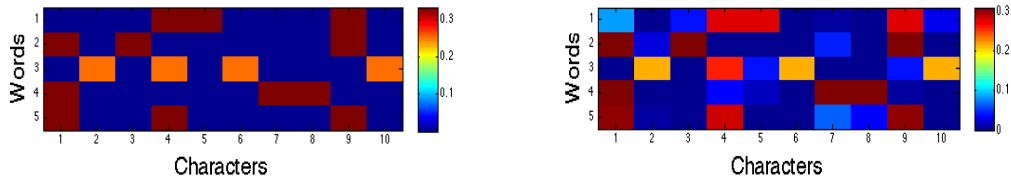

Figure 6: Oracle Experiment 1 emission distribution (L) True distribution; (R) Learnt distribution

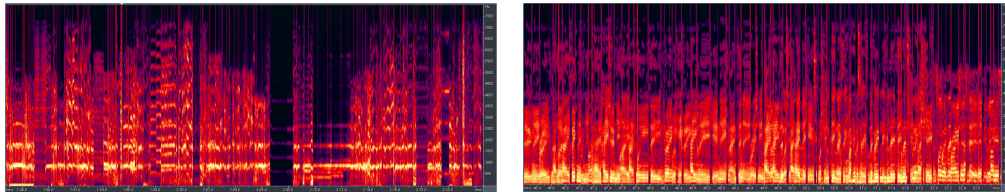

Figure 7: Instances of log-spectra for 2 AUDs, with all occurrences across files concatenated. (L) Predominantly music; (R) Predominantly speech. The y-axis correspond to frequency bins

## 5 Experiments

In this section, we present a pair of oracle experiments to verify that the segmentation model performs as expected. Then, we present results using the 2-level hierarchical model on the event kit of the 2011 TRECVID Multimedia Event Detection (MED) task [22].

**Oracle Experiment 1**: We picked five words {the, cat, ate, blue, man} and used our model to generate 100 documents ($s \sim \mathcal{N}(1.5, 0.2)$), and use this as the data to learn words unsupervised. The results of learning are sensitive to initialization as is expected with EM-based algorithms, but end up fairly close to the true parameters. The learned words are distributions over characters, and we would expect the algorithm to recover a distribution close to the true distribution. Figure 6 compares the true and learned distributions over characters for the five words. We notice that the learned one is not as sparse, but the higher-probability characters are very similar. To quantify the error, the average error per parameter is 0.003 averaged over 5 random initializations.

**Oracle Experiment 2**: We performed a similar experiment with web URLs concatenated together, since web URLs have a clear structure beginning with "http://" or "www" and containing ".com". Again, the learner automatically identified the most frequent word to be one which had highest emission probabilities for {'.', 'c', 'o', 'm'} and the second most frequent word with {'h', 't', 'p', '/', ':', 'w'} characters having high probabilities. The respective segments identified for those words while decoding conforms to our expectations and usually corresponded to ".com" and "http://www".

**TRECVID 2011 MED data**: The MED data was used for our experiments, and contains 15 semantically defined categories (*e.g.* board trick, feeding an animal– full list at [22]), and was to be used to build detectors for each semantic category, so that given a new file, it can predict whether it belongs to any of those categories or not. All our reported results use 8-fold cross validation on the entire event kit. Performance is evaluated using the Area Under the Missed Detection Rate against False Alarm Rate curve (AUC, henceforth) for each category (hence, the lower the better).

The entire data was used to learn a lexicon of AUDs, and each file was decoded using the models. Figure 7 shows two examples of log-spectra of all frames spanned by an AUD concatenated together. In both cases, one can see structural consistency, showing that the AUDs find acoustically similar segments as desired. While the AUDs are not required to have clear semantic interpretations, listening to the concatenated instances shows that the AUD on the left primarily spans music segments while the right consists primarily of speech– speech formant structures are visible in the image.

We then use the decoded AUD sequences as character streams to learn parameters for the second layer of *observable acoustic events* spanning local AUD sequences. Based on the learnt models, we can decode the data in terms of these *events*. Since there are no annotations available, these events

Table 1: Performance summary across MED11 dataset (lower is better)

| System | VQ | FOLEY | AUD-FREQ | EVENT-FREQ | COMB |
|---|---|---|---|---|---|
| Average AUC | 0.2971 | 0.2624 | 0.2174 | 0.2297 | **0.1842** |
| Best Performance in #categories | 0 | 0 | 2 | 1 | 12 |
| #SSI-over-VQ | 0 | 5 | 15 | 12 | 15 |
| #SSI-over-FOLEY | 0 | 0 | 15 | 11 | 15 |
| #SSI-over-AUD-FREQ | 0 | 0 | 0 | 2 | 12 |
| #SSI-over-EVENT-FREQ | 0 | 0 | 11 | 0 | 15 |

are not assigned semantics, but listening to multiple instances concatenated together shows similar phenomena being captured. One such event consists of sequences of sounds that relate to crowds with loud cheering and a babble of voices in a party being subsumed within the same *event*.

We use the decoded AUDs and event sequences for each file to characterize the MED11 data, and evaluate the effect of using the AUDs layer and the event layer individually (AUD-FREQ and EVENT-FREQ, respectively) and together (COMB). For the AUD-FREQ characterization, each file is represented by a $k$-dimensional feature vector (one for each of the $k$ AUDs in the vocabulary, with the frequency of occurrence of the AUD in the file being the feature value). The EVENT-FREQ characterization is similarly set up while COMB is a concatenation of the two.

We compare our models with two baselines that are commonly used in such tasks. The first is a VQ baseline (VQ) where a set of audio words is learned unsupervised by applying K-Means on the data at the frame level. The second uses an audio library to create a supervised sound library from the 480 sound types in the Foley Sound Library [23], and we characterize each file using occurrence information of these sounds in the file (FOLEY).

These feature representations were used to train a random forest classifier [24] with 500 trees for each class. Table 1 summarizes the performance of the various feature settings on the MED11 data (lower AUC is better). We used the best performing lexicon size for the various systems– 4096 clusters for the VQ, 480 Foley audio events, 1024 AUDs, and 128 acoustic events. The AUC numbers reported are averaged across all 15 categories in the dataset. The last 4 rows of the column indicate how feature sets compare pairwise by noting the number of categories in which one improves over the other with statistical significance ($p < 0.05$, in a paired $t$-test). For some events, the performance of different feature settings is not statistically significantly different. We observe that, overall, the AUD-FREQ feature set seems to perform the best by itself, but combining it with the second layer of events in our hierarchical framework results in the best overall system on average AUC, and the combination outperforms AUDs alone on 12 of the 15 categories with statistical significance.

# 6   Conclusion and Future Work

Although the results above only show gains obtained in objective evaluations on a standard large-scale retrieval tasks, the "events" discovered by the learning algorithm have deeper significance– they represent automatically learned characterizations of longer-scale acoustic phenomena with semantic import. This work presents an initial approach to extracting such deeper semantic features from audio based on local patterns of low-level acoustic units. There are a few directions we hope to explore in the future. Since the discovered latent events and acoustic units do not have true labels, we would like to explore ways to leverage tags, knowledge bases and human annotators to induce labels. In such settings, we would like to explore non-parametric techniques that can grow the event set based on data. The distribution characteristics here are simple unigrams without additional structure; we would like to explore other models with appropriate priors for each layer in the hierarchical model. Finally, we would like to use such event structure to study co-occurrences and dependencies of acoustic event types that might allow us to predict sounds in the future based on the context.

**Acknowledgments**

This work was supported by funding from Charles Stark Draper Labs, Cambridge, USA.

## Footnotes

[1]We do not permit length 0 in our experiments, instead forcing a minimum length

# References

[1] E. Wold, T. Blum, D. Keislar, and J.W. Wheaton. Content-based classication, search, and retrieval of audio. *IEEE Multimedia*, 3:27–36, 1996.

[2] A.G. Hauptmann and M.J. Witbrock. Informedia: News-on-demand multimedia information acquisition and retrieval. In *Proceedings of Intelligent Multimedia Information Retrieval*, pages 213–239. AAAI Press, 1997.

[3] G. Guo and S.Z. Li. Content-based audio classication and retrieval by support vector machines. *IEEE Transactions on Neural Nets*, 14, 2003.

[4] M. Slaney. Mixture of probability experts for audio retrieval and indexing. In *Proceedings of the International Conference of Multimedia and Expo*, 2002.

[5] M. Slaney. Semantic audio retrieval. In *Proceedings of the International Conference on Acoustic Speech and Signal Processing*, 2002.

[6] S.F. Chang, D. Ellis, W. Jiang, K. Lee, A. Yanagawa, A. Loui, and J. Luo. Large-scale multimodal semantic concept detection for consumer video. In *Proceedings of the MIR workshop, ACM-Multimedia*, 2007.

[7] S. Sundaram and S. Narayanan. Classication of sound clips by two schemes: using onomatopoeia and semantic labels. In *Proceedings of the IEEE International Conference of Multimedia and Expo*, 2008.

[8] Z. Liu, J. Huang, and Y. Wang. Classification of tv programs based on audio information using hidden markov model. In *Proceedings of the 2nd IEEE Workshop on Multimedia Signal Processing*, 1998.

[9] S. Berrani, G. Manson, and P. Lechat. A non-supervised approach for repeated sequence detection in tv broadcast streams. In *Signal Processing: Image Communication*, volume 23, pages 525–537, 2008.

[10] G. Friedland, L. Gottlieb, and A. Janin. Using artistic markers and speaker identification for narrative-theme navigation of seinfeld episodes. In *Workshop on Content-Based Audio/Video Analysis for Novel TV Services, 11th IEEE International Symposium on Multimedia*, 2009.

[11] S. Kim, S. Sundaram, P. Georgiou, and S. Narayanan. Audio scene understanding using topic models. In *Proceedings of the NIPS Workshop on Applications for Topic Models: Text and Beyond*, 2009.

[12] S. Kim, S. Sundaram, P. Georgiou, and S. Narayanan. Acoustic stopwords for unstructured audio information retrieval. In *Proceedings of the 18th European Signal Processing Conference*, 2010.

[13] X. Zhu. Semi-supervised learning with graphs. *PhD Thesis*, 2005.

[14] S. Chaudhuri, M. Harvilla, and B. Raj. Unsupervised learning of acoustic unit descriptors for audio content representation and classification. In *Proceedings of Interspeech*, 2011.

[15] W. Li. Random texts exhibit zipf's-law-like word frequency distribution. *IEEE Transactions on Information Theory*, 38:1842–1845, 1992.

[16] J. Eeckhout. Gibrat's law for (all) cities. *American Economic Review*, 94:1429–1451, 2004.

[17] D. Mochihashi, T. Yamada, and N. Ueda. Bayesian unsupervised word segmentation with nested pitman-yor language modeling. In *Proceedings of the 47th Meeting of the Association for Computational Linguistics*, 2009.

[18] H. Poon, C. Cherry, and K. Toutanova. Unsupervised morphological segmentation with log-linear models. In *Proceedings of the 47th Meeting of the Association for Computational Linguistics*, 2009.

[19] S. Goldwater, T.L. Griffiths, and M. Johnson. A bayesian framework for word segmentation: Exploring the effects of context. *Cognition*, 112:21–54, 2009.

[20] M. Johnson and S. Goldwater. Improving nonparametric bayesian inference: Experiments on unsupervised word segmentation with adaptor grammars. In *Proceedings of Human Language Technologies: North American Chapter of the Association for Computational Linguistics*, 2009.

[21] A. P. Dempster, N.M. Laird, and D.B. Rubin. Maximum likelihood from incomplete data via the EM algorithm. *Journal of the Royal Statistical Society, Series B*, 39:1–38, 1977.

[22] TRECVID Multimedia Event Detection Task. http://www.nist.gov/itl/iad/mig/med11.cfm. 2011.

[23] The Art of Foley. http://www.sound-ideas.com/artfoley.html. 2005.

[24] L. Breiman. Random forests. *Machine Learning*, 45:5–32, 2001.

